# Learning by Combining Memorization and Gradient Descent

**John C. Platt**
Synaptics, Inc.
2860 Zanker Road, Suite 206
San Jose, CA 95134

## ABSTRACT

We have created a radial basis function network that allocates a new computational unit whenever an unusual pattern is presented to the network. The network learns by allocating new units and adjusting the parameters of existing units. If the network performs poorly on a presented pattern, then a new unit is allocated which memorizes the response to the presented pattern. If the network performs well on a presented pattern, then the network parameters are updated using standard LMS gradient descent. For predicting the Mackey Glass chaotic time series, our network learns much faster than do those using back-propagation and uses a comparable number of synapses.

## 1  INTRODUCTION

Currently, networks that perform function interpolation tend to fall into one of two categories: networks that use gradient descent for learning (e.g., back-propagation), and constructive networks that use memorization for learning (e.g., k-nearest neighbors).

Networks that use gradient descent for learning tend to form very compact representations, but use many learning cycles to find that representation. Networks that memorize their inputs need to only be exposed to examples once, but grow linearly in the training set size.

The network presented here strikes a compromise between memorization and gradient descent. It uses gradient descent for the "easy" input vectors and memorization for the "hard" input vectors. If the network performs well on a particular input

vector, or the particular input vector is already close to a stored vector, then the network adjusts its parameters using gradient descent. Otherwise, it memorizes the input vector and the corresponding output vector by allocating a new unit. The explicit storage of an input-output pair means that this pair can be used immediately to improve the performance of the system, instead of merely using that information for gradient descent.

The network, called the resource-allocation network (RAN), uses units whose response is localized in input space. A unit with a non-local response needs to undergo gradient descent, because it has a non-zero output for a large fraction of the training data.

Because RAN is a constructive network, it automatically adjusts the number of units to reflect the complexity of the function that is being interpolated. Fixed-size networks either use too few units, in which case the network memorizes poorly, or too many, in which case the network generalizes poorly. Parzen windows and K-nearest neighbors both require a number of stored patterns that grow linearly with the number of presented patterns. With RAN, the number of stored patterns grows sublinearly, and eventually reaches a maximum.

## 1.1   PREVIOUS WORK

Previous workers have used networks with localized basis functions (Broomhead & Lowe, 1988) (Moody & Darken, 1988 & 89) (Poggio & Girosi, 1990). Moody has further extended his work by incorporating a hash table lookup (Moody, 1989). The hash table is a resource-allocating network where the values in the hash table only become non-zero if the entry in the hash table is activated by the corresponding presence of non-zero input probability.

The RAN adjusts the centers of the Gaussian units based on the error at the output, like (Poggio & Girosi, 1990). Networks with centers placed on a high-dimensional grid, such as (Broomhead & Lowe, 1988) and (Moody, 1989), or networks that use unsupervised clustering for center placement, such as (Moody & Darken, 1988 & 89) generate larger networks than RAN, because they cannot move the centers to increase the accuracy.

Previous workers have created function interpolation networks that allocate fewer units than the size of training set. Cascade-correlation (Fahlman & Lebiere, 1990), SONN (Tenorio & Lee, 1989), and MARS (Friedman, 1988) all construct networks by adding additional units. These algorithms work well. The RAN algorithm improves on these algorithms by making the addition of a unit as simple as possible. RAN uses simple algebra to find the parameters of a new unit, while cascade-correlation and MARS use gradient descent and SONN uses simulated annealing.

## 2   THE ALGORITHM

This section describes a resource-allocating network (RAN), which consists of a network, a strategy for allocating new units, and a learning rule for refining the network.

## 2.1   THE NETWORK

The RAN is a two-layer radial-basis-function network. The first layer consists of

units that respond to only a local region of the space of input values. The second layer linearly aggregates outputs from these units and creates the function that approximates the input-output mapping over the entire space.

A simple function that implements a locally tuned unit is a Gaussian:

$$z_j = \sum_k (c_{jk} - I_k)^2,$$
$$x_j = \exp(-z_j / w_j^2). \tag{1}$$

We use a $C^1$ continuous polynomial approximation to speed up the algorithm, without loss of network accuracy:

$$x_j = \begin{cases} \left(1 - (z_j / q w_j^2)\right)^2, & \text{if } z_j < q w_j^2; \\ 0, & \text{otherwise;} \end{cases} \tag{2}$$

where $q = 2.67$ is chosen empirically to make the best fit to a Gaussian.

Each output of the network $y_i$ is a sum of the outputs $x_j$, each weighted by the synaptic strength $h_{ij}$ plus a global polynomial. The $x_j$ represent information about local parts of the space, while the polynomial represents global information:

$$y_i = \sum_j h_{ij} x_j + \sum_k L_{ik} I_k + \gamma_i. \tag{3}$$

The $h_{ij} x_j$ term can be thought of as a bump that is added or subtracted to the polynomial term $\sum_k L_{ik} I_k + \gamma_i$ to yield the desired function.

The linear term is useful when the function has a strong linear component. In the results section, the Mackey-Glass equation was predicted with only a constant term.

## 2.2   THE LEARNING ALGORITHM

The network starts with a blank slate: no patterns are yet stored. As patterns are presented to it, the network chooses to store some of them. At any given point the network has a current state, which reflects the patterns that have been stored previously.

The allocator may allocate a new unit to memorize a pattern. After the new unit is allocated, the network output is equal to the desired output $\vec{T}$. Let the index of this new unit be $n$.

The peak of the response of the newly allocated unit is set to the memorized input vector,

$$\vec{c}_n = \vec{I}. \tag{4}$$

The linear synapses on the second layer are set to the difference between the output of the network and the novel output,

$$\vec{h}_n = \vec{T} - \vec{y}. \tag{5}$$

The width of the response of the new unit is proportional to the distance from the nearest stored vector to the novel input vector,

$$w_n = \kappa ||\vec{I} - \vec{c}_{\text{nearest}}||, \tag{6}$$

where $\kappa$ is an overlap factor. As $\kappa$ grows larger, the responses of the units overlap more and more.

The RAN uses a two-part memorization condition. An input-output pair $(\vec{I}, \vec{T})$ should be memorized if the input is far away from existing centers,

$$||\vec{I} - \vec{c}_{\text{nearest}}|| > \delta(t), \tag{7}$$

and if the difference between the desired output and the output of the network is large

$$||\vec{T} - \vec{y}(\vec{I})|| > \epsilon. \tag{8}$$

Typically, $\epsilon$ is a desired accuracy of output of the network. Errors larger than $\epsilon$ are immediately corrected by the allocation of a new unit, while errors smaller than $\epsilon$ are gradually repaired using gradient descent. The distance $\delta(t)$ is the scale of resolution that the network is fitting at the $t$th input presentation. The learning starts with $\delta(t) = \delta_{\max}$, which is the largest length scale of interest, typically the size of the entire input space of non-zero probability density. The distance $\delta(t)$ shrinks until the it reaches $\delta_{\min}$, which is the smallest length scale of interest. The network will average over features that are smaller than $\delta_{\min}$. We used a function:

$$\delta(t) = \max(\delta_{\max} \exp(-t/\tau), \delta_{\min}), \tag{9}$$

where $\tau$ is a decay constant.

At first, the system creates a coarse representation of the function, then refines the representation by allocating units with smaller and smaller widths. Finally, when the system has learned the entire function to the desired accuracy and length scale, it stops allocating new units altogether.

The two-part memorization condition is necessary for creating a compact network. If only condition (7) is used, then the network will allocate units instead of using gradient descent to correct small errors. If only condition (8) is used, then fine-scale units may be allocated in order to represent coarse-scale features, which is wasteful.

By allocating new units the RAN eventually represents the desired function ever more closely as the network is trained. Fewer units are needed for a given accuracy if the first-layer synapses $c_{jk}$, the second-level synapses $h_{ij}$, and the parameters for the global polynomial $\gamma_i$ and $L_{ik}$ are adjusted to decrease the error: $\mathcal{E} = ||\vec{y} - \vec{T}||^2$ (Widrow & Hoff, 1960). We use gradient descent on the second-layer synapses to decrease the error whenever a new unit is not allocated:

$$\begin{aligned} \Delta h_{ij} &= \alpha(T_i - y_i)x_j, \\ \Delta \gamma_i &= \alpha(T_i - y_i), \\ \Delta L_{ik} &= \alpha(T_i - y_i)I_k. \end{aligned} \tag{10}$$

In addition, we adjust the centers of the responses of units to decrease the error:

$$\Delta c_{jk} = 2\frac{\alpha}{w_j}(I_k - c_{jk})x_j \left[\sum_i (T_i - y_i)h_{ij}\right]. \qquad (11)$$

Equation (11) is derived from gradient descent and equation (1). Empirically, equation (11) also works for the polynomial approximation (2).

## 3   RESULTS

One application of an interpolating RAN is to predict complex time series. As a test case, a chaotic time series can be generated with a nonlinear algebraic or differential equation. Such a series has some short-range time coherence, but long-term prediction is very difficult.

The RAN was tested on a particular chaotic time series created by the Mackey-Glass delay-difference equation:

$$x(t + 1) = (1 - b)x(t) + a\frac{x(t - \tau)}{1 + x(t - \tau)^{10}}, \qquad (12)$$

for $a = 0.2$, $b = 0.1$, and $\tau = 17$. We trained the network to predict the value $x(T + \Delta T)$, given the values $x(T), x(T - 6), x(T - 12)$, and $x(T - 18)$ as inputs.

The network was tested using two different learning modes: off-line learning with a limited amount of data, and on-line learning with a large amount of data. The Mackey-Glass equation has been learned off-line, by other workers, using the back-propagation algorithm (Lapedes & Farber, 1987), and radial basis functions (Moody & Darken, 1989). We used RAN to predict the Mackey-Glass equations with the following parameters: $\alpha = 0.02$, 400 learning epochs, $\delta_{max} = 0.7$, $\kappa = 0.87$ and $\delta_{min} = 0.07$ reached after 100 epochs. RAN was simulated using $\epsilon = 0.02$ and $\epsilon = 0.05$. In all cases, $\Delta T = 85$.

Figure 1 shows the efficiency of the various learning algorithms: the smallest, most accurate algorithms are towards the lower left. When optimized for size of network ($\epsilon = 0.05$), the RAN has about as many weights as back-propagation and is just as accurate. The efficiency of RAN is roughly the same as back-propagation, but requires much less computation: RAN takes approximately 8 minutes of SUN-4 CPU time to reach the accuracy listed in figure 4, while back-propagation took approximately 30–60 minutes of Cray X-MP time.

The Mackey-Glass equation has been learned using on-line techniques by hashing B-splines (Moody, 1989). We used on-line RAN using the following parameters: $\alpha = 0.05$, $\delta_{max} = 0.7$, $\delta_{min} = 0.07$, $\kappa = 0.87$, and $\delta_{min}$ reached after 5000 input presentations. Table 1 compares the on-line error versus the size of network for both RAN and the hashing B-spline (Moody, personal communication). In both cases, $\Delta T = 50$. The RAN algorithm has similar accuracy to the hashing B-splines, but the number of units allocated is between a factor of 2 and 8 smaller.

For more detailed results on the Mackey-Glass equation, see (Platt, 1991).

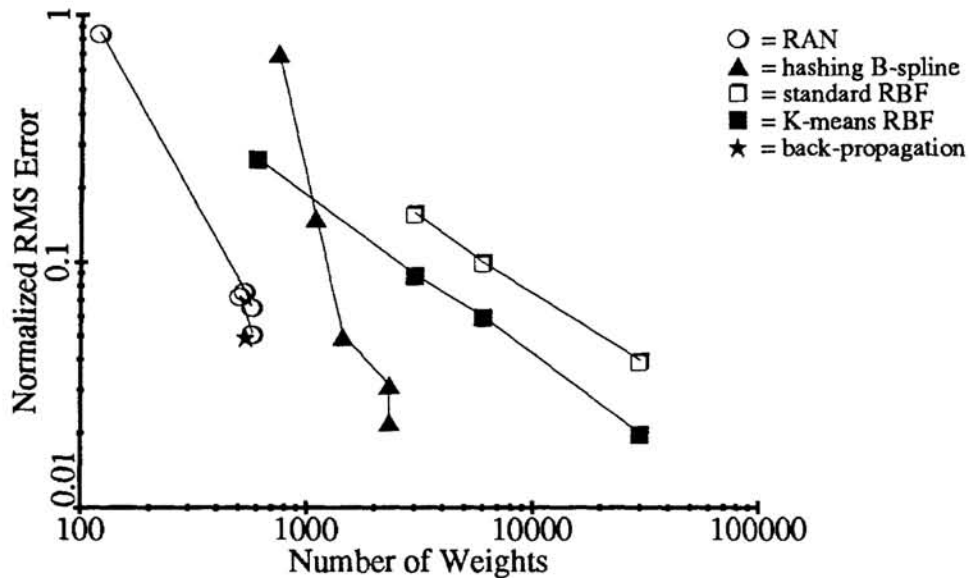

**Figure 1:** The error on a test set versus the size of the network. Back-propagation stores the prediction function very compactly and accurately, but takes a large amount of computation to form the compact representation. RAN is as compact and accurate as back-propagation, but uses much less computation to form its representation.

**Table 1:** Comparison between RAN and hashing B-splines

| Method | Number of Units | Normalized RMS Error |
|---|---|---|
| RAN, $\epsilon = 0.05$ | 50 | 0.071 |
| RAN, $\epsilon = 0.02$ | 143 | 0.054 |
| Hashing B-spline 1 level of hierarchy | 284 | 0.074 |
| Hashing B-spline 2 levels of hierarchy | 1166 | 0.044 |

## 4   CONCLUSIONS

There are various desirable attributes for a network that learns: it should learn quickly, it should learn accurately, and it should form a compact representation. Formation of a compact representation is particularly important for networks that are implemented in hardware, because silicon area is at a premium. A compact representation is also important for statistical reasons: a network that has too many parameters can overfit data and generalize poorly.

Many previous network algorithms either learned quickly at the expense of a compact representation, or formed a compact representation only after laborious computation. The RAN is a network that can find a compact representation with a reasonable amount of computation.

**Acknowledgements**

Thanks to Carver Mead, Carl Ruoff, and Fernando Pineda for useful comments on the paper. Special thanks to John Moody who not only provided useful comments on the paper, but also provided data on the hashing B-splines.

**References**

Broomhead, D., Lowe, D., 1988, Multivariable function interpolation and adaptive networks, *Complex Systems,* **2,** 321–355.

Fahlman, S. E., Lebiere, C., 1990, The Cascade-Correlation Learning Architecture, *In:* Advances in Neural Information Processing Systems 2, D. Touretzky, ed., 524–532, Morgan-Kaufmann, San Mateo.

Friedman, J. H., 1988, Multivariate Adaptive Regression Splines, Department of Statistics, Stanford University, Tech. Report LCS102.

Lapedes, A., Farber, R., 1987, *Nonlinear Signal Processing Using Neural Networks: Prediction and System Modeling,* Technical Report LA-UR-87-2662, Los Alamos National Laboratory, Los Alamos, NM.

Moody, J, Darken, C., 1988, Learning with Localized Receptive Fields, *In:* Proceedings of the 1988 Connectionist Models Summer School, D. Touretzky, G. Hinton, T. Sejnowski, eds., 133–143, Morgan-Kaufmann, San Mateo.

Moody, J, Darken, C., 1989, Fast Learning in Networks of Locally-Tuned Processing Units, *Neural Computation,* **1(2),** 281–294.

Moody, J., 1989, Fast Learning in Multi-Resolution Hierarchies, *In:* Advances in Neural Information Processing Systems 1, D. Touretzky, ed., 29–39, Morgan-Kaufmann, San Mateo.

Platt., J., 1991, A Resource-Allocating Network for Function Interpolation, *Neural Computation,* **3(2),** to appear.

Poggio, T., Girosi, F., 1990, Regularization Algorithms for Learning that are Equivalent to Multilayer Networks, *Science,* **247,** 978–982.

Powell, M. J. D., 1987, Radial Basis Functions for Multivariable Interpolation: A Review, *In:* Algorithms for Approximation, J. C. Mason, M. G. Cox, eds., Clarendon Press, Oxford.

Tenorio, M. F., Lee, W., 1989, Self-Organizing Neural Networks for the Identification Problem, *In:* Advances in Neural Information Processing Systems 1, D. Touretzky, ed., 57–64, Morgan-Kaufmann, San Mateo.

Widrow, B., Hoff, M., 1960, Adaptive Switching Circuits, *In:* 1960 IRE WESCON Convention Record, 96–104, IRE, New York.